# STATISTICAL PREDICTION WITH KANERVA'S SPARSE DISTRIBUTED MEMORY

David Rogers
Research Institute for Advanced Computer Science
MS 230-5, NASA Ames Research Center
Moffett Field, CA 94035

## ABSTRACT

A new viewpoint of the processing performed by Kanerva's sparse distributed memory (SDM) is presented. In conditions of near- or over- capacity, where the associative-memory behavior of the model breaks down, the processing performed by the model can be interpreted as that of a *statistical predictor*. Mathematical results are presented which serve as the framework for a new statistical viewpoint of sparse distributed memory and for which the standard formulation of SDM is a special case. This viewpoint suggests possible enhancements to the SDM model, including a procedure for improving the predictiveness of the system based on Holland's work with 'Genetic Algorithms', and a method for improving the capacity of SDM even when used as an associative memory.

## OVERVIEW

This work is the result of studies involving two seemingly separate topics that proved to share a common framework. The first topic, *statistical prediction*, is the task of associating extremely large perceptual state vectors with future events. The second topic, *over-capacity in Kanerva's sparse distributed memory* (SDM), is a study of the computation done in an SDM when presented with many more associations than its stated capacity.

I propose that in conditions of over-capacity, where the associative-memory behavior of an SDM breaks down, the processing performed by the SDM can be used for statistical prediction. A mathematical study of the prediction problem suggests a variant of the standard SDM architecture. This variant not only behaves as a statistical predictor when the SDM is filled beyond capacity but is shown to double the capacity of an SDM when used as an associative memory.

## THE PREDICTION PROBLEM

The earliest living creatures had an ability, albeit limited, to *perceive* the world through crude senses. This ability allowed them to react to changing conditions in

the environment; for example, to move towards (or away from) light sources. As nervous systems developed, learning was possible; if food appeared simultaneously with some other perception, perhaps some odor, a creature could learn to associate that smell with food.

As the creatures evolved further, a more rewarding type of learning was possible. Some perceptions, such as the perception of pain or the discovery of food, are very important to an animal. However, by the time the perception occurs, damage may already be done, or an opportunity for gain missed. *If a creature could learn to associate current perceptions with future ones, it would have a much better chance to do something about it before damage occurs.* This is the *prediction problem.*

The difficulty of the prediction problem is in the extremely large number of possible sensory inputs. For example, a simple animal might have the equivalent of 1000 bits of sensory data at a given time; in this case, the number of possible inputs is greater than the number of atoms in the known universe! In essence, it is an enormous search problem: a living creature must find the subregions of the perceptual space which correlate with the features of interest. Most of the gigantic perceptual space will be uncorrelated, and hence uninteresting.

## THE OVERCAPACITY PROBLEM

An associative memory is a memory that can recall data when addressed 'close-to' an address where data were previously stored. A number of designs for associative memories have been proposed, such as Hopfield networks (Hopfield, 1986) or the nearest-neighbor associative memory of Baum, Moody, and Wilczek (1987). Memory-related standards such as capacity are usually selected to judge the relative performance of different models. Performance is severely degraded when these memories are filled beyond capacity.

Kanerva's sparse distributed memory is an associative memory model developed from the mathematics of high-dimensional spaces (Kanerva, 1988) and is related to the work of David Marr (1969) and James Albus (1971) on the cerebellum of the brain. (For a detailed comparison of SDM to random-access memory, to the cerebellum, and to neural-networks, see (Rogers, 1988b)). Like other associative memory models, it exhibits non-memory behavior when near- or over- capacity.

Studies of capacity are often over-simplified by the common assumption of *uncorrelated random* addresses and data. The capacity of some of these memories, including SDM, is degraded if the memory is presented with correlated addresses and data. Such correlations are likely if the addresses and data are from a real-world source. Thus, understanding the over-capacity behavior of an SDM may lead to better procedures for storing correlated data in an associative memory.

# SPARSE DISTRIBUTED MEMORY

Sparse distributed memory can be best illustrated as a variant of random-access memory (RAM). The structure of a twelve-location SDM with ten-bit addresses and ten-bit data is shown in figure 1. (Kanerva, 1988)

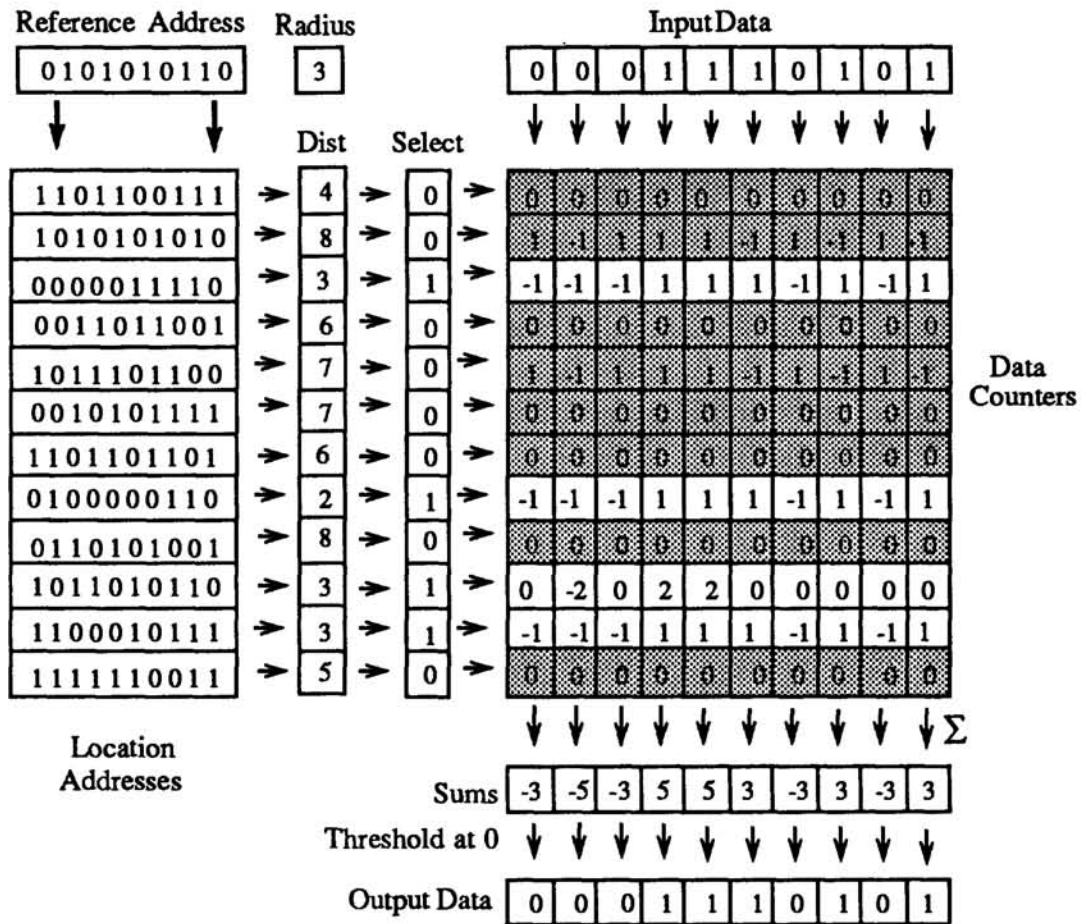

**Figure 1.** Structure of a Sparse Distributed Memory

A *memory location* is a row in this figure. The *location addresses* are set to random addresses. The *data counters* are initialized to zero. All operations begin with *addressing* the memory; this entails finding the Hamming distance between the *reference address* and each of the location addresses. If this distance is less than or equal to the *Hamming radius*, the select-vector entry is set to 1, and that location is termed *selected*. The ensemble of such selected locations is called the *selected set*. Selection is noted in the figure as non-gray rows. A *radius* is chosen so that only a small percentage of the memory locations are selected for a given reference address.

(Later, we will refer to the fact that a memory location defines an *activation set* of addresses in the address space; the activation set corresponding to a location is the set of reference addresses which activate that memory location. Note the reciprocity

between the selected set corresponding to a given reference address, and the activation set corresponding to a given location.)

When *writing* to the memory, all selected counters beneath elements of the *input data* equal to 1 are incremented, and all selected counters beneath elements of the *input data* equal to 0 are decremented. This completes a write operation. When *reading* from the memory, the selected data counters are summed columnwise into the register *sums*. If the value of a sum is greater than or equal to zero, we set the corresponding bit in the *output data* to 1; otherwise, we set the bit in the *output data* to 0. (When reading, the contents of the *input data* are ignored.)

This example makes clear that a datum is *distributed* over the data counters of the selected locations when writing, and that the datum is reconstructed during reading by *averaging* the sums of these counters. However, depending on what additional data were written into some of the selected locations, and depending on how these data correlate with the original data, the reconstruction may contain noise.

## THE BEHAVIOR OF AN SDM WHEN AT OVER-CAPACITY

Consider an SDM with a 1,000-bit address and a 1-bit datum. In this memory, we are storing associations that are samples of some binary function **f** on the space **S** of all possible addresses. After storing only a few associations, each data counter will have no explicit meaning, since the data values stored in the memory are distributed over many locations. However, once a sufficiently large number of associations are stored in the memory, the data counter gains meaning: when appropriately normalized to the interval [0, 1], it contains a value which is the *conditional probability that the data bit is* 1, *given that its location was selected.* This is shown in figure 2.

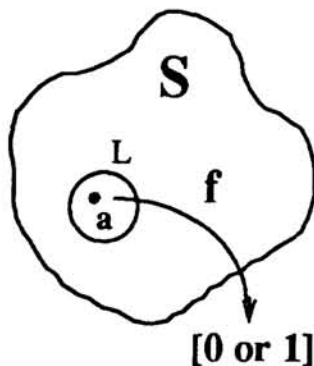

• S is the space of all possible addresses

• L is the set of addresses in S which activate a given memory location

• f is a binary function on S that we want to estimate using the memory

• The data counter for L contains the average value of f over L, which equals $P( f(X) = 1 \mid X \in L )$

**[0 or 1]**

**Figure 2.** The Normalized Content of a Data Counter is the Conditional Probability of the Value of **f** Being Equal to 1 Given the Reference Addresses are Restricted to the Sphere L.

In the prediction problem, we want to find activation sets of the address space that correlate with some desired feature bit. *When filled far beyond capacity, the indi-*

*vidual memory locations of an SDM are collecting statistics about individual subregions of the address space.* To estimate the value of **f** at a given address, it should be possible to combine the conditional probabilities in the data counters of the selected memory locations to make a "best guess".

In the prediction problem, **S** is the space of possible sensory inputs. Since most regions of **S** have no relationship with the datum we wish to predict, most of the memory locations will be in non-informative regions of the address space. Associative memories are not useful for the prediction problem because the key part of the problem is the search for subregions of the address space that are informative. Due to capacity limitations and the extreme size of the address space, memories fill to capacity and fail before enough samples can be written to identify the useful subregions.

# PREDICTING THE VALUE OF f

Each data counter in an SDM can be viewed as an independent estimate of the conditional probability of **f** being equal to 1 over the activation set defined by the counter's memory location. If a point of **S** is contained in multiple activation sets, each with its own probability estimate, how do we combine these estimates? More directly, when does knowledge of membership in some activation set help us estimate **f** better?

Assume that we know $P(f(X) = 1)$, which is the average value of **f** over the entire space **S**. If a data counter in memory location L has the same conditional probability as $P(f(X) = 1)$, then knowing an address is contained in the activation set defining L *gives no additional information.* (This is what makes the prediction problem hard: most activation sets in S will be uncorrelated with the desired datum.)

When is a data counter useful? If a data counter contains a conditional probability far away from the probability for the entire space, then it is highly informative. The more committed a data counter is one way or the other, the more weight it should be given. Ambivalent data counters should be given less weight.

Figure 3 illustrates this point. Two activation sets of S are shown; the numbers 0 and 1 are the values of **f** at points in these sets. (Assume that *all* the points in the activation sets are in these diagrams.) Membership in the left activation set is non-informative, while membership in the right activation set is highly informative. Most activation sets are neither as bad as the left example nor as good as the right example; instead, they are intermediate to these two cases. We can calculate the relative weights of different activation sets if we can estimate the relative signal/noise ratio of the sets.

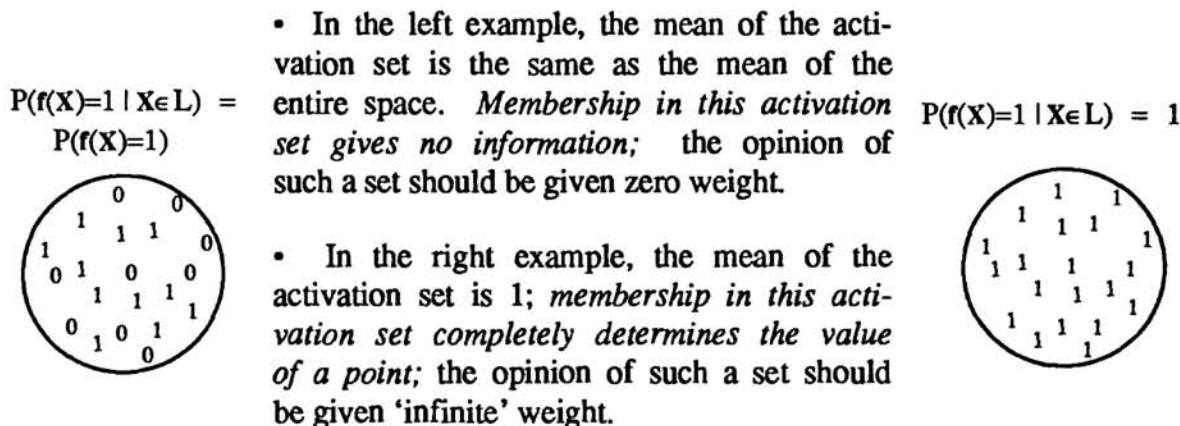

- In the left example, the mean of the activation set is the same as the mean of the entire space. *Membership in this activation set gives no information;* the opinion of such a set should be given zero weight.

- In the right example, the mean of the activation set is 1; *membership in this activation set completely determines the value of a point;* the opinion of such a set should be given 'infinite' weight.

**Figure 3.** The Predictive Value of an Activation Set Depends on How Much New Information it Gives About the Function **f**.

To obtain a measure of the amount of signal in an activation set $L$, imagine segregating the points of $L$ into two sectors, which I call the *informative sector* and the *non-informative sector.* (Note that this partition will not be unique.) Include in the non-informative sector the largest number of points possible such that the percentage of 1's and 0's equals the corresponding percentages in the overall population of the entire space. The remaining points, which constitute the informative sector, will contain all 0's or 1's. The relative size $r$ of the informative sector compared to $L$ constitutes a measure of the signal. The relative size of the non-informative sector to $L$ is $(1 - r)$, and is a measure of the noise. Such a conceptual partition is shown in figure 4.

Once the signal and the noise of an activation set is estimated, there are known methods for calculating the weight that should be given to this set when combining with other sets (Rogers, 1988a). That weight is $(r / (1 - r)^2)$. Thus, given the conditional probability and the global probability, we can calculate the weight which should be given to that data counter when combined with other counters.

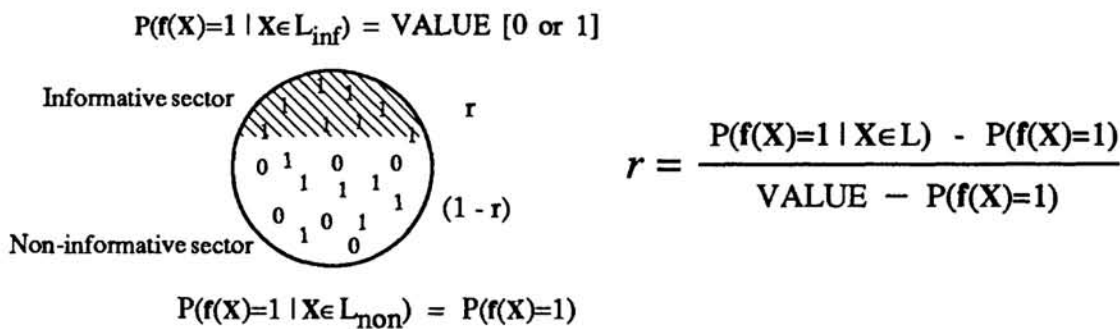

**Figure 4.** An Activation Set Defined by a Memory Location can be Partitioned into Informative and Non-informative Sectors.

## EXPERIMENTAL

The given weighting scheme was used in the standard SDM to test its effect on capacity. In the case of random addresses and data, the weights doubled the capacity of the SDM. Even greater savings are likely with correlated data. These results are shown in figure 5.

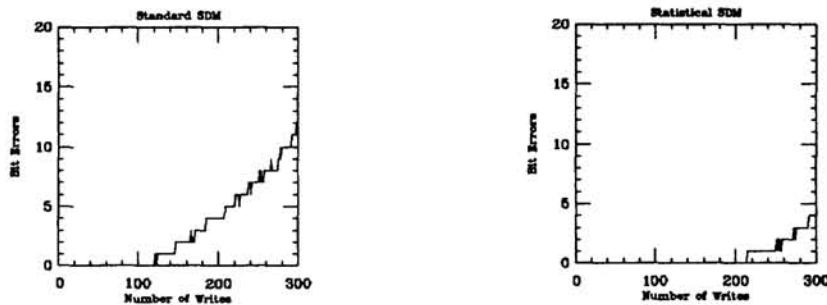

**Figure 5.** Number of Bitwise Errors vs. Number of Writes in a 256-bit Address, 256-bit Data, 1000-Location Sparse Distributed Memory. The Left is the Standard SDM; the Right is the Statistically-Weighted SDM. Graphs Shown are Averages of 16 Runs

In deriving the weights, it was assumed that the individual data counters would become meaningful only when a sufficiently large number of associations were stored in the memory. This experiment suggests that even a small number of associations is sufficient to benefit from statistically-based weighting. These results are important, for they suggest that this scheme can be used in an SDM in the full continuum, from low-capacity memory-based uses to over-capacity statistical-prediction uses.

## CONCLUSIONS

Studies of SDM under conditions of over-capacity, in combination with the new problem of statistical prediction, suggests a new range of uses for SDM. By weighting the locations differently depending on their contents, we also have discovered a technique for improving the capacity of the SDM even when used as a memory.

This weighting scheme opens new possibilities for learning; for example, these weights can be used to estimate the fitness of the locations for learning algorithms such as Holland's *genetic algorithms*. Since the statistical prediction problem is primarily a problem of search over extremely large address spaces, such techniques would allow redistribution of the memory locations to regions of the address space which are maximally useful, while abandoning the regions which are non-informative. The combination of *learning* with *memory* is a potentially rich area for future study.

Finally, many studies of associative memories have explicitly assumed random data

in their studies; most real-world applications have non-random data. This theory explicitly assumes, and makes use of, correlations between the associations given to the memory. Assumptions such as randomness, which are useful in mathematical studies, must be abandoned if we are to apply these tools to real-world problems.

## Acknowledgments

This work was supported in part by Cooperative Agreements NCC 2-408 and NCC 2-387 from the National Aeronautics and Space Administration (NASA) to the Universities Space Research Association (USRA). Funding related to the Connection Machine was jointly provided by NASA and the Defense Advanced Research Projects Agency (DARPA). All agencies involved were very helpful in promoting this work, for which I am grateful.

The entire RIACS staff and the SDM group has been supportive of my work. Louis Jaeckel gave important assistance which guided the early development of these ideas. Bruno Olshausen was a vital sounding-board for this work. Finally, I'll get mushy and thank those who supported my spirits during this project, especially Pentti Kanerva, Rick Claeys, John Bogan, and last but of course not least, my parents, Philip and Cecilia. Love you all.

## References

Albus, J. S., "A theory of cerebellar functions," *Math. Bio.*, 10, pp. 25-61 (1971).

Baum, E., Moody, J., and Wilczek, F., "Internal representations for associative memory," *Biological Cybernetics*, (1987).

Holland, J. H., *Adaptation in natural and artificial systems*, Ann Arbor: University of Michigan Press (1975).

Holland, J. H., "Escaping brittleness: the possibilities of general-purpose learning algorithms applied to parallel rule-based systems," in *Machine learning, an artificial intelligence approach, Volume II*, R. J. Michalski, J. G. Carbonell, and T. M. Mitchell, eds. Los Altos, California: Morgan Kaufmann (1986).

Hopfield, J.J., "Neural networks and physical systems with emergent collective computational abilities," *Proc. Nat'l Acad. Sci. USA*, 79, pp. 2554-8 (1982).

Kanerva, Pentti., "Self-propagating Search: A Unified Theory of Memory," Center for the Study of Language and Information Report No. CSLI-84-7 (1984).

Kanerva, Pentti., *Sparse distributed memory*, Cambridge, Mass: MIT Press, 1988.

Marr, D., "The cortex of the cerebellum," *J. Physio.*, 202, pp. 437-470 (1969).

Rogers, David, "Using data-tagging to improve the performance of Kanerva's sparse distributed memory," Research Institute for Advanced Computer Science Technical Report 88.1, NASA Ames Research Center (1988a).

Rogers, David, "Kanerva's sparse distributed memory: an associative memory algorithm well-suited to the Connection Machine," Research Institute for Advanced Computer Science Technical Report 88.32, NASA Ames Research Center (1988b).
